# Learning with Target Prior

**Zuoguan Wang**
Dept. of ECSE, Rensselaer Polytechnic Inst.
Troy, NY 12180
wangz6@rpi.edu

**Siwei Lyu**
Computer Science, Univ. at Albany, SUNY
Albany, NY 12222
lsw@cs.albany.edu

**Gerwin Schalk**
Wadsworth Center, NYS Dept. of Health
Albany, NY, 12201
schalk@wadsworth.org

**Qiang Ji**
Dept. of ECSE, Rensselaer Polytechnic Inst.
Troy, NY 12180
jiq@rpi.edu

## Abstract

In the conventional approaches for supervised parametric learning, relations between data and target variables are provided through training sets consisting of pairs of corresponded data and target variables. In this work, we describe a new learning scheme for parametric learning, in which the target variables $\mathbf{y}$ can be modeled with a prior model $p(\mathbf{y})$ and the relations between data and target variables are estimated with $p(\mathbf{y})$ and a set of uncorresponded data $\mathbf{X}$ in training. We term this method as *learning with target priors* (LTP). Specifically, LTP learning seeks parameter $\theta$ that maximizes the log likelihood of $f_\theta(\mathbf{X})$ on a uncorresponded training set with regards to $p(\mathbf{y})$. Compared to the conventional (semi)supervised learning approach, LTP can make efficient use of prior knowledge of the target variables in the form of probabilistic distributions, and thus removes/reduces the reliance on training data in learning. Compared to the Bayesian approach, the learned parametric regressor in LTP can be more efficiently implemented and deployed in tasks where running efficiency is critical. We demonstrate the effectiveness of the proposed approach on parametric regression tasks for BCI signal decoding and pose estimation from video.

## 1 Introduction

One of the central problems in machine learning is prediction/inference, where given an input datum $\mathbf{X}$, we would like to predict or infer the value of a target variable of interest, $\mathbf{y}$, assuming $\mathbf{X}$ and $\mathbf{y}$ have some intrinsic relationship. The prediction/inference task in many practical applications involves high dimensional and structured data and target variables. Depending on the form of knowledge about $\mathbf{X}$ and $\mathbf{y}$ and their relationship available to us, there are several different methodologies to solve the prediction inference problem.

In the Bayesian approach, our knowledge about input and target variables, as well as their relationships, are all represented as probability distributions. Correspondingly, the prediction/inference task is solved with optimizations based on the posterior distribution $p(\mathbf{y}|\mathbf{X})$, a common choice of which is the *maximum a posteriori* objective: $\max_{\mathbf{y}} p(\mathbf{y}|\mathbf{X})$. The posterior distribution can be explicitly constructed from the target prior, $p(\mathbf{y})$, which encodes our knowledge on the internal structure of the target $\mathbf{y}$, and the likelihood, $p(\mathbf{X}|\mathbf{y})$, which summarizes the process of generating $\mathbf{X}$ from $\mathbf{y}$, as $p(\mathbf{y}|\mathbf{X}) \propto p(\mathbf{X}|\mathbf{y})p(\mathbf{y})$. Or it can be directly handled as in the conditional random fields [9] without referring to the target prior or the likelihood. The advantage of the Bayesian approach is that it incorporates prior knowledge about data and target variables into the prediction/inference task in a principled manner. The main downside is that, in many practical problems, the relationship between $\mathbf{X}$ and $\mathbf{y}$ could be complicated and defy straightforward modeling. Furthermore, except for a few special cases (e.g., Gaussian models), the Bayesian prediction/inference of $\mathbf{y}$ from data $\mathbf{X}$ usually requires expensive numerical optimization or Monte-Carlo sampling.

An alternative approach to prediction/inference is supervised parametric learning, where the information about $\mathbf{X}$ and $\mathbf{y}$ and their relationship is described in the form of a set of corresponding examples, $\{\mathbf{X}_i, \mathbf{y}_i\}_{i=1}^m$, and the goal of learning is to choose an optimal member from a parametric family $f_\theta(\mathbf{X})$ that minimizes the average prediction error using a loss function $\min_\theta \frac{1}{m} \sum_{i=1}^m L(\mathbf{y}_i - f_\theta(\mathbf{X}_i))$. Usually, the optimization may also include a regularization penalty on $\theta$ to reduce over-fitting. The most significant drawback of the supervised parametric learning approach is that the learning performance relies heavily on the quality and quantity of the training data. This problem is somewhat alleviated in semi-supervised learning [28], where the training data include unlabeled examples of $\mathbf{X}$. However, unlike the Bayesian approach, it is usually difficult to incorporate prior knowledge in the form of probabilistic distributions into (semi)supervised parametric learning.

In this work, we describe a new approach to learning a parametric regressor $f_\theta(\mathbf{X})$, which we term as *learning with target prior* (LTP). In many practical applications, the target variables $\mathbf{y}$ follow the some regular spatial and temporal patterns that can be described probabilistically, and the observed target variables are samples of such distributions. For instance, to perform an activity like grasping a cup, the traces of finger movements tend to have similar patterns that are caused by many factors, such as the underlying physiological, anatomical and dynamic constraints. Such regular patterns can benefit the task of decoding the finger movements from ECoG signals in a brain computer interface (BCI) system, Fig.1, as it regularizes the decoder to produce similar patterns. Similarly, the skeleton structures and the dynamic dependencies constraint the body pose to have similar spatial and temporal patterns for the same activity (e.g. walking, running and jumping), which can be used for body pose estimation in computer vision.

In LTP learning, we incorporate such spatial and temporal regular patterns of the target variables into the learning framework. Specifically, we learn a probability distribution $p(\mathbf{y})$ that captures the spatial and temporal regularities of the target variable $\mathbf{y}$, then we estimate the function parameters $\theta$, by maximizing the log-likelihood of the output $y = f_\theta(\mathbf{X})$ with respect to the the prior distribution. LTP learning can be applied to both unsupervised learning, in which no corresponded input and output are available, and semi-supervised learning in which part of corresponding outputs are available. We demonstrate the effectiveness of LTP learning in two problems: BCI decoding and pose estimation.

The rest of the paper is organized as the following: Section 2 discusses the related work. Section 3 describes the general framework for our method and compare with other existing methodologies. In Sections 4 and 5, details on deployment and experimental evaluation of this general framework in two applications, namely BCI decoding and pose estimation from video, are described. Section 6 concludes the paper with discussion and future works.

## 2   Related Work

LTP learning is related to several existing learning schemes. The prior knowledge about the target variables in classification problems is exploited in recent works as learning with uncertain labels, in which the distribution over the target class labels for each data example is used in place of corresponding pairs of data/target variables [10]. A similar idea in semi-supervised learning uses the proportion of different classes [16, 28] to predict the class labels on the uncorresponded training data examples. The knowledge about class proportion conditioned on certain input feature is captured by generalized expectation (GE)[12, 13]. There are several works directly embed domain constraints about the target variables in learning. For instance, constraint driven learning (CODL) [3] enforces task specific constraints on the target labels by appending a penalty term in the objective function. Posterior regularization [5] directly imposes regularization on the posterior of the latent target variables, of which CODL can be seen as a special case with MAP approximation. A general framework, which incorporates prior information as measurements in the Bayesian framework, is proposed in [11]. However, all these approaches have only been applied to problems with discrete outputs (classification or labeling) and may be difficult to extend to incorporate complex dependencies in high-dimensional continuous target variables.

LTP learning is also related to learning with structured outputs. Dependencies in the target variables can be directly modeled in conditional random fields (CRF) [9], as a probabilistic graphical model between the output components. However, the learned regressor is usually not in closed form and predictions has to be obtained by numerical optimization or Monte-Carlo sampling. Some of the recent supervised parametric learning methods can take advantage of some structure constraints over the target variables. The max margin Markov network [21] trains an SVM classifier with outputs

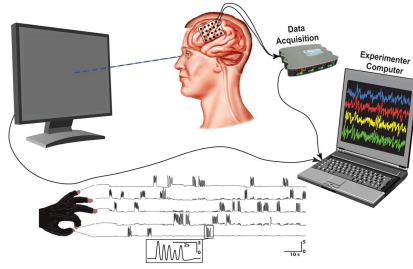

Figure 1: *Experiment setup for this study.*

whose structures are described by graphs. The structured SVM was further extended with high order loss function [20] or models with latent variables [27]. These methods can be viewed as special cases of LTP learning, where general probabilistic models for target variables can be incorporated.

## 3 General Framework

In this section, we describe the general framework of learning with target priors. Specifically, our task is to learn the parameter $\theta$ in a parametric family of functions of $\mathbf{X}$, $f_\theta(x)$, to best predict the corresponding target variable $\mathbf{y}$. Both the data and target variable can be of high dimensions. Knowledge about target variable is provided through a target prior model in the form of a parametric probability distribution, $p_\eta(\mathbf{y})$, with model parameter $\eta$. The specific form of $p_\eta(\mathbf{y})$ is determined based on different applications, ranging from simple distributions to more complex models such as Markov random fields. The model parameter $\theta$ is estimated by maximizing the log-likelihood $\log p_\eta(f_\theta(\mathbf{X}))$. In the following, we apply the LTP learning to unsupervised learning in which no corresponded input and output are available, as well as semi-supervised learning in which part of corresponding outputs are available.

For the unsupervised learning, assume we are given a set of outputs $\mathbf{y} \in \mathcal{R}^{\mathcal{Y} \times m}$, as well as a set of uncorresponded inputs $\mathbf{X} \in \mathcal{R}^{\mathcal{X} \times n}$, where $\mathcal{Y}$ and $\mathcal{X}$ are the dimensionality, and $m$ and $n$ are the temporal length for $\mathbf{y}$ and $\mathbf{X}$ respectively. This is applicable to the case of BCI where it is easier to gather inputs $\mathbf{X}$ or structured targets $\mathbf{y}$ than it is to gather corresponded inputs and targets $(\mathbf{X}, \mathbf{y})$. In many real BCI applications the input brain signals $\mathbf{X}$ are collected only under thoughts without actual body movement $\mathbf{y}$. The body movements could be easily collected when the brain signals are not being recorded. In the problem of pose estimation, it is a tedious work to label poses $\mathbf{y}$ on the input images $\mathbf{X}$. In both the finger movement decoding and pose estimation, $\mathbf{y}$ and $\mathbf{X}$ could be extracted from different subjects. A prior model $p_\eta(\mathbf{y})$ is learned from $\{\mathbf{y}_i\}_{i=1}^m$, where $\mathbf{y}_i \in \mathcal{R}^{\mathcal{Y} \times 1}$ and $\eta$ is parameter of the prior model. Then the function parameter $\theta$ is estimated by maximizing

$$\max_\theta \frac{1}{n} \sum_{i=1}^n \log p_\eta(f_\theta(\mathbf{X}_i)), \tag{1}$$

where $\mathbf{X}_i \in \mathcal{R}^{\mathcal{X} \times 1}$. The parameter $\theta$ is chosen in the way that the output on the $\{\mathbf{X}_i\}_{i=1}^n$ maximally consistent of the prior distribution $p_\eta(\mathbf{y})$. The setting of semi-supervised learning is slightly different from unsupervised learning, in which the corresponding input $\{\mathbf{X}_i\}_{i=1}^m$ of the output $\{\mathbf{y}_i\}_{i=1}^m$ are also given. Then the learning becomes the combination of supervised and unsupervised learning:

$$\min_\theta \frac{1}{m} \sum_{i=1}^m L(\mathbf{y}_i - f_\theta(\mathbf{X}_i)) - \frac{\lambda}{n} \sum_{i=1}^n \log p_\eta(f_\theta(\mathbf{X}_i)), \tag{2}$$

where $L$ is the loss function and $\lambda$ is a constant representing the tradeoff between the two terms. In eq. 2, the parameter $\theta$ is chosen in the way that the outputs not only minimize the loss function on training data, but also make the predicted targets on the unlabeled data comply with the target prior.

Next, we adapt unsupervised/semi-supervised learning with LTP to the prediction/inference in two applications, namely, decoding ECoG signal to predict finger movement in BCI and estimation of body poses from videos, where the-state-of-the-art performances are achieved.

## 4 Finger Movement Decoding in ECoG based BCI

The main task in brain-computer interface (BCI) systems is to convert electronic signals recorded from human brain into controlling commands for patients with motor disabilities (e.g., paralysis). Many recent studies in neurobiology have suggested that electrocorticographic (ECoG) signals

recorded near the brain surface show strong correlations with limb motions [2, 8]. ECoG signal decoding is the critical step in ECoG based BCI systems, the goal of which is to obtain a functional mapping between the ECoG signals and the kinematic variables (e.g., spatial locations and movement velocities of fingers recorded by a digital glove) [8]. The ECoG decoding problem has been widely solved with supervised parametric learning [26, 8, 25], where corresponded ECoG signals and target kinematic variables are collected from one subject and used to train a parametric regressor. However, the decoder learned from data collected from one subject in a controlled experiment usually has trouble to generalize for the same subject over time and in an open environment (temporal generalization) [18], or to decode signals from other subjects (cross-subject generalization) [24]. The former is due to the strong variances in ECoG signals that are caused by other concurrent brain activities, and the latter is due to the difference in shape and volume of the brains for different subjects. These limitations are regarded as the most challenging issues in current BCI systems [7].

There have been several works addressing these issues. For instance, to improve the generalization performance across trials, several adaptive classification methods are proposed [18], i.e., updating the LDA with labeled feed back data. To generalize better across subjects, a collaborative paradigm was proposed to integrating information from multiple subjects [24]. In [17] it is investigated that certain spectral features of ECoG signals can be used across subjects to classify movements. However, these methods do not provide satisfactory solutions since the central challenge in extending the parametric decoder across time and subject is that the conventional parametric learning approach, on which all these methods are based, relies on training data to obtain information for learning the regressor, which in these cases are difficult to collect. At the same time, in BCI it is typically much easier to gather samples of uncorresponded target variables, i.e, traces of finger movements recorded by digital gloves, than it is to gather corresponding pairs of training samples.

Thus in this work, we propose to improve the temporal and cross-subject generalization of BCI decoders with the learning with target priors framework. In the first step, we obtain a parametric target prior model using uncorresponded samples of the target data, in this case, the traces of finger positions. In the second step, we estimate a linear decoding function using the general method described in Section 3. Let us first define notations that are to be used subsequently: we use a linear decoding function, as: $f_\theta(x) = \mathbf{X}^T\theta$, to predict the traces of finger movements $\mathbf{y}$ as target variable. Specifically, we define $\mathbf{y} \in \mathcal{R}^{\mathcal{Y}}$ where $\mathcal{Y}$ corresponds to the number of samples in the finger traces. $\mathbf{X} \in \mathcal{R}^{L \times \mathcal{Y}}$ is a matrix whose columns are a subset of ECoG signal features of length $L$. The model parameter $\theta \in \mathcal{R}^L$ is a vector. Linear decoding function are widely used in BCI decoding [1] for its simplicity and run-time efficiency in constructing hardware based BCI system.

## 4.1 Target Prior Model

We use the *Gaussian-Bernoulli restricted Boltzmann machine* (GB-RBM) [14]: $p_\eta(\mathbf{y}) = \frac{1}{Z}\sum_\mathbf{h} e^{-E_\eta(\mathbf{y},\mathbf{h})}$, where $Z$ is the normalizing constant, and $\mathbf{h} \in \{0,1\}^{\mathcal{H}}$ are binary hidden variables, as the parametric target prior model. The pdf is defined in terms of the joint energy function over $\mathbf{y}$ and $\mathbf{h}$, as:

$$E_\eta(\mathbf{y}, \mathbf{h}) = \sum_{i=1}^{\mathcal{Y}} \frac{(\mathbf{y}_i - \mathbf{c}_i)^2}{2} - \sum_{i=1,j=1}^{\mathcal{Y},\mathcal{H}} \mathbf{W}_{ij}\mathbf{y}_i\mathbf{h}_j - \sum_{j=1}^{\mathcal{H}} \mathbf{b}_j\mathbf{h}_j.$$

where $\mathbf{W}_{ij}$ is the interaction strength between the hidden node $\mathbf{h}_i$ and visible node $\mathbf{y}_j$. $\mathbf{c}$ and $\mathbf{b}$ are the bias for the visible layer and hidden layer, respectively. The target variable $\mathbf{y}$ is normalized to have zero mean and unit standard variance. The parameters in this model, $(\mathbf{W}, \mathbf{c}, \mathbf{b})$, are collectively represented with $\eta$. Direct maximum likelihood training of GB-RBM is intractable due to the normalizing factor $Z$, so we use contrastive divergence [6] to estimate $\eta$ from data.

## 4.2 Learning Regressor Parameter $\theta$

With training data and the GB-RBM as the target prior model, we optimize the objective function of LTP in Eq.(1) or (2) for parameters $\theta$. With the linear decoding function and squared loss function, the gradient of the first term of Eq.(2) can be computed as $-\frac{2}{m}\sum_{i=1}^m \mathbf{X}_i(\mathbf{y}_i - \mathbf{X}_i^T\theta)$. The derivative of $\theta$ over log-likelihood of $\mathbf{X}^T\theta$ with regards to the prior model can be computed, as

$$\frac{\partial \log p_\eta(\mathbf{X}^T\theta)}{\partial \theta} = \sum_\mathbf{h} p_\eta(\mathbf{h}|\mathbf{X}^T\theta)\frac{-\partial E(\mathbf{X}^T\theta, \mathbf{h})}{\partial \theta}. \tag{3}$$

Plugging the energy function $E$ into Eq.(3), we can simplify it to

$$\frac{\partial \log p_\eta(\mathbf{X}^T\theta)}{\partial \theta} = \mathbf{X}(\mathbf{X}^T\theta - \mathbf{c}) + \mathbf{XW}^T \sum_{\mathbf{h}} p_\eta(\mathbf{h}|\mathbf{X}^T\theta)\mathbf{h}, \qquad (4)$$

where $\sum_{\mathbf{h}} p_\eta(\mathbf{h}|\mathbf{X}^T\theta)\mathbf{h}$ using the property of GB-RBM that the elements of $\mathbf{h}$ are independent given $\mathbf{X}^T\theta$. Specifically, assume $\mathbf{g} = \sum_{\mathbf{h}} p_\eta(\mathbf{h}|\mathbf{X}^T\theta)\mathbf{h}$, then $\mathbf{g}_i = \sigma(\mathbf{W}_i\mathbf{X}^T\theta)$, where $\mathbf{W}_i$ is the $i$th row of $\mathbf{W}$ and $\sigma$ is the logistic function $\sigma(x) = 1/(1 + \exp(-x))$. The expectation of the derivative over all sequences, composed of $\mathcal{Y}$ successive samples in the training data, can be expressed as $\langle \frac{\partial \log p_\eta(\mathbf{X}^T\theta)}{\partial \theta} \rangle_{data}$ where $< \cdot >_{data}$ stands for expectation over the data.

### 4.3 Experimental Settings

The ECoG data and target finger movement variables are collected from a clinical setting based on five subjects (A-E) who underwent brain surgeries [8]. Each subject had a $48$- or $64$- electrode grid placed over the cortex. During the experiment, the subjects are required to repeatedly flex and extend specific individual fingers according to visual cues on a video screen. The experiment setup is shown in Fig. 1. The data collection for each subject lasted 10 minutes, which yielded an average of 30 trials for each finger. The flexion of each finger was measured by a data glove. For each channel, features are extracted based on signal power of three bands (1-60Hz, 60-100Hz, 100-200Hz) [2], which results in 144 or 204 features for subjects with 48 or 64 channels, respectively.

### 4.4 Learning Target Prior Model and Decoding Function

The training data for the prior model $p_\eta(\mathbf{y})$ are either from other subjects or from the same subject but were collected at a different time and do not have correspondence with the training input data. Here we consider the finger moving traces only composed of flexion and extension as in Fig. 2(A). This simplified model is still practically useful since we can first classify the trace into movement state or rest state and then apply the corresponding regressor for each state [4]. Each subject has around 1400 samples. We model the finger movement trace using the GB-RBM with $64$ hidden nodes and 12 visible nodes, which is approximately the length of one round extension and flexion. Then, all segments from 12 successive samples in the data are used to train the prior model.

The GB-RBM is trained with stochastic gradient decent with a mini-batch size of 25 sub-sequences. We run 5000 epochs with a fixed learning rate 0.001. We first validate the prior model by drawing samples from the learned GB-RBM. Figure 2(B) shows the 9 samples, which seem to capture some important properties of the temporal dynamics of the finger trace.

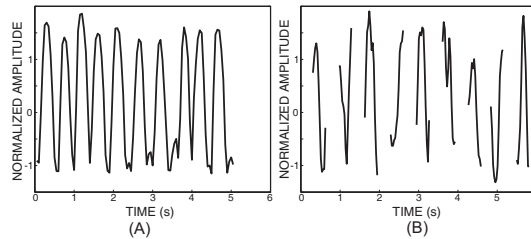

Figure 2: *(A) Original trace; (B)samples from GB-RBM. Each sample is a segment with length 12.*

With the prior model, the paired features/target variables if they exist and unpaired features, on which the expectation of Eq.(4) is calculated, are used to learn the parameter $\theta$. $\theta$ is randomly initialized and learned with stochastic gradient decent with the same batch size 25. We run 2000 epochs with fixed learning rate $10^{-4}$.

### 4.5 Generalization Across Subjects

We learn the decoding function for new subjects by deploying the unsupervised LTP learning in Section 3. Even though it is difficult to get the corresponded samples from new subjects, we always have the input ECoG signals, whose features will be used as the input of the unsupervised LTP learning.

We compare the unsupervised LTP learning with linear regression [2] in two ways: 1) the linear regression (intra subject) in which the corresponded data and target variables are available. The accuracy of linear regression is calculated based on five fold cross-validation, that is, 4/5 trials (25 trials) are used for training and 1/5 trials (5 trials) are used for testing. 2) the linear regression (inter

Table 1: *Results on thumb of subjects based on 2 fold cross validation (correlation coefficient).*

|  | A | B | C | D | E |
|---|---|---|---|---|---|
| Linear R | 0.29 | 0.26 | 0.06 | 0.10 | 0.11 |
| Semisupervised LTP | 0.38 | 0.42 | 0.13 | 0.15 | 0.12 |

subject) trained on the one subject and tested on other subjects. The results for inter subjects are calculated based on 5 fold cross-validation (each time one subject is used for training and the model is tested on other four subjects). Linear regression is trained on pairs of features and targets while LTP only uses the targets to train the prior model. For the linear regression trained and tested on different subjects, the channels across subjects are aligned by the 3-d position of the sensors.

Figure 3(A) shows the performance comparison of the three models. Note that the performances of the unsupervised LTP learning is on par with those of the linear regression (intra) on subject A, B, C and D, which suggests that the decoder learned by unsupervised LTP learning can generalize across subjects. Figure 3(B) and (C) shows two examples of prediction results from the unsupervised LTP learning. On the other hand, not surprisingly, the performances of linear regression (inter subjects) suggest that it cannot be extended across subjects, which is due to brain difference for different subjects as stated above. The generalization ability gained by unsupervised LTP learning is mainly because it directly learns decoding functions on the new subject without using brain signal from existing subjects, which are believed to change dramatically among subjects. One thing we noticed is that the unsupervised LTP learning does not work well on subject E, which is because the thumb movement speed of subject E is much slower than subject A, on which the prior model is trained. This suggests that the quality of the target prior model is critical for the performance.

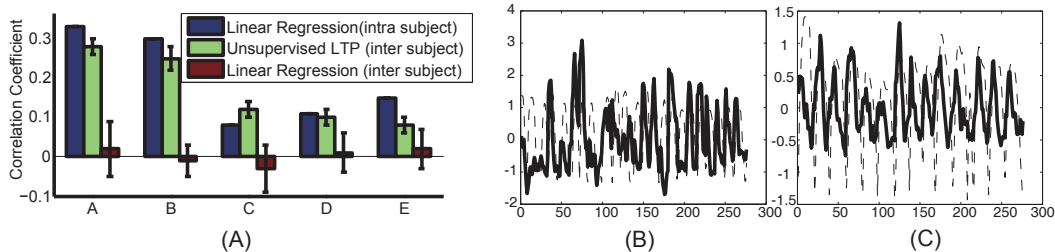

Figure 3: *(A) Comparison among three models across subjects; (B) Sample results for subject A; (C) Sample results for subject B. The dot line is the ground truth and the solid line is the prediction*

## 4.6   Online Learning for Decoding Functions

In the next set of experiment, we use the learning with target priors framework for learning decoding functions that generalize over time. This experiment is performed for each subject individually. For each subject, assume $\{\mathbf{X}_i, \mathbf{y}_i\}_{i=1}^m$ be the training data in the current trial and $\{\mathbf{X}_j\}_{j=1}^n$ be the new samples unknown in training. We first train the prior model on $\{\mathbf{y}_i\}_{i=1}^m$. Then parameter $\theta$ is learned using the semi-supervised learning in section 3.

The new samples come sequentially and thus we want the decoding function to be online updated. The parameter $\theta$ is not updated for every new coming sample, but every batch of data $\mathbf{X} \in R^{L \times \mathcal{Y}}$. Here we give a brief description of the online batch updating method. For the start, the parameter $\theta$ is learned from the corresponding pairs of samples $\{\mathbf{X}_i, \mathbf{y}_i\}_{i=1}^m$. Then the decoding function with parameter $\theta$ is used to decode the first batch $\{\mathbf{X}_j\}_{j=1}^{\mathcal{Y}}$. After the batch $\{\mathbf{X}_j\}_{i=1}^{\mathcal{Y}}$ is decoded, $\{\mathbf{X}_j\}_{j=1}^{\mathcal{Y}}$, not including the predicted target variables, is included as part of the unlabeled training data to update the parameter $\theta$ by the semi-supervised learning in section 3. Then the updated $\theta$ is used to decode the second batch $\{\mathbf{X}_j\}_{j=\mathcal{Y}+1}^{2\mathcal{Y}}$ and the process loops. In summary, after the new coming batch is decoded using the current parameter $\theta$, then it is included as training data to update parameter $\theta$. Generally, we are trying to maximally use the "seen" data to get the decoding function prepared for the "unseen" coming samples.

The batch size $\mathcal{Y}$ is chosen to be 12. The model is tested on the thumb of five subjects based on 2 fold cross validation, that is, we treat the first 15 trials as the paired data/target variables and then online test the remaining trials. After that in turn we treat the last 15 trials as the paired data/target variables and use the first 15 trials for online testing. The results in Table 1 show the proposed model with online batch updating can significantly improve the results. This means that by regularizing

the new features with the target prior, the semi-supervised learning in Section 3 successfully obtains information from the new features and adapts the decoders well for new coming samples.

## 5 Pose Estimation from Videos

In this section, we apply learning with target priors to the problem of the pose estimation problem, the goal of which is to extract 3D human pose from images or video sequences. We demonstrate LTP by applying it to learn a linear mapping from image features to poses while LTP could be used to learn more sophisticated models. We will show that the algorithms learned by LTP are more generalizable both across subjects and over time on the same subject respectively.

In this experiment, we use six walking sequences from CMU MoCap database (http://mocap.cs.cmu.edu). The data are from 3 subjects, with sequences 1 & 2 from the first subject, sequences 3 & 4 from the second subject and sequences 5 & 6 from the third subject. Each sequence consists of about 70 frames. Our task is to estimate the 3-D pose from videos, which is described by 59 dimensional joint angles. The image feature is extracted from the silhouette image at the side view. For each silhouette image we take 10 dimension moment features [23].

We represent the video sequence as $\{\mathbf{X}_i, \mathbf{y}_i\}_{i=1}^n$, where $\mathbf{X} \in R^{10 \times n}$ are the image features, $\mathbf{y} \in R^{59 \times n}$ are the joint angles, where $n$ is the length of the sequence and $n$ could be different for different sequences. Instead of directly mapping features to 59 dimensional joint angles, we learn the function which maps the features to the 3 dimensional subspace of joint angles obtained through PCA. Then the original space of joint angles is recovered from the low dimensional subspace. All

---

**Algorithm 1** learning with target priors

   **Input:** joint angles $\{\mathbf{y}_i\}_{i=1}^n$, test features $\mathbf{X}^*$
   **Output:** $\mathbf{y}^*$ corresponding to $\mathbf{X}^*$
   **Steps:**
   **1:** PCA: $\mathbf{y} \rightarrow EZ$, where $E \in R^{59 \times 3}$, $Z \in R^{3 \times n}$
   **2:** learn prior model $p_\eta(\mathbf{y})$ on $Z$
   **3:** learn mapping function $Z^* = f_\theta(X^*)$ using the unsupervised LTP learning in section 3
   **Output:** recover original space $\mathbf{y}^* = EZ^*$

---

possible segments composed of successive 60 frames in the sequence are used to train the GB-RBM. So the length of the vector into the GB-RBM is 180 (the subspace is 3 dimension).

Many methods have been proposed to address the pose estimation problem, among which sGPLVM [19], FOLS-GPLVM [15] and imCRBM [22] are the three very competitive ones. sGPLVM models a shared latent space by pose and image features through GPLVM, while FOLS-GPLVM models a shared latent space and a private latent space for each part. imCRBM constructs a pose prior for the Bayesian model using the implicit mixture of CRBM. However, Taylor's work is not comparable to our method, because it requires a generative model to directly map a pose to a silhouette, while our method explicitly uses the extracted moment features, and the comparison here focuses on algorithms instead of features. So we will compare with sGPLVM and FOLS-GPLVM using the same image features. The training of both sGPLVM and FOLS-GPLVM require corresponded images and poses $(\mathbf{X}, \mathbf{y})$ while LTP does not require this.

For the unsupervised LTP learning, the target prior model is trained on the subspace of the joint angles $\{\mathbf{y}_i\}_{i=1}^n$ on sequence 1 and tested on the features of all 6 sequences. The implementation details are shown in algorithm 1. Except for sGPLVM and FOLS-GPLVM, the results are also compared with ridge regression. Ridge regression, sGPLVM and FOLS-GPLVM are trained on the first sequence with paired samples $\{\mathbf{X}_i, \mathbf{y}_i\}_{i=1}^n$ and tested on all the 6 sequences. The implementation of ridge regression is similar to that in algorithm 1, the only difference is that the mapping from features to the PCA subspace is through ridge regression.

The results are measured in terms of mean absolute joint angle error and are shown in table 2. We can see that when testing on the sequence from the same subject (sequence 2), unsupervised LTP learning is not the best. In contrast, when testing on the sequences from subjects B and C, unsupervised LTP learning achieves the best results, which is slightly better than sGPLVM. Considering that only linear dimension reduction and linear function are assumed for unsupervised LTP learning and paired samples are not required, unsupervised LTP learning is even more competitive. FOLS-GPLVM does not perform well on this data set, which is probably due to limited training samples. Thus the experiments demonstrate that the algorithm learned by unsupervised

Table 2: *Train prior model on the first sequence and test on all sequences. Results are measured with mean absolute joint angle error.*

| Subject | A | | B | | C | |
|---|---|---|---|---|---|---|
| Sequence Num | 1 | 2 | 3 | 4 | 5 | 6 |
| Ridge Regression | **2.1** | 4.8 | 8.3 | 8.5 | 10.7 | 10.7 |
| sGPLVM | — | **3.1** | 5.6 | 6.1 | 3.0 | 3.1 |
| FOLS-GPLVM | — | 5.3 | 6.5 | 6.4 | 3.3 | 4.0 |
| Unsupervised LTP | 3.0 | 4.8 | **5.3** | **6.1** | **2.9** | **2.9** |

Table 3: *For each subject, train on the first sequence and test on the second sequence. Results are measured with absolute joint angle error.*

| Subject | A | B | C |
|---|---|---|---|
| Ridge Regression | 4.8 | 5.3 | 3.1 |
| sGPLVM | 3.1 | 5.3 | 3.0 |
| FOLS-GPLVM | 5.3 | 5.8 | 3.8 |
| Semi-supervised LTP | **2.87** | **3.97** | **2.33** |

LTP learning in section 3 can generalize well across subjects. The reason that ridge regression, sGPLVM and FOLS-GPLVM do not generalize well is that the relations between poses and images are solely learned from corresponded poses and images, and these relations may have difficulty to hold for the new subjects due to may factors (i.e, the video for the new subject is recorded from a slightly different angle). LTP avoids this problem by learning the relations using the generalizable prior distribution over the targets and the images from the new subjects.

We further demonstrate that the algorithm learned through semi-supervised learning in section 3 generalizes well across time for the same subject. In this experiment, for each subject we treat the first sequence as the paired samples $\{\mathbf{X}_i, \mathbf{y}_i\}_{i=1}^m$ and estimate the 3-D pose of the second sequence $\{\mathbf{X}_i\}_{j=1}^n$. The prior model is trained on the joint angles of the first sequence $\{\mathbf{y}_i\}_{i=1}^m$. The algorithm is similar to that in algorithm 1 except for replacing unsupervised LTP learning with semi-supervised learning. The results in table 3 show that the semi-supervised learning in section 3 significantly outperforms three other methods.

## 6    Conclusion and Discussion

In this work, we describe a new learning scheme for parametric learning, known as *learning with target priors*, that uses a prior model over the target variables and a set of uncorresponded data in training. Compared to the conventional (semi)supervised learning approach, LTP can make efficient use of prior knowledge of the target variables in the form of probabilistic distributions, and thus removes/reduces the reliance on training data in learning. Compared to the Bayesian approach, the learned parametric regressor in LTP can be more efficiently implemented and deployed in tasks where running efficiency is critical, such as on-line BCI signal decoding. We demonstrate the effectiveness of the proposed approach in terms of generalization on parametric regression tasks for BCI signal decoding and pose estimation from video.

There are several extensions of this work we would like to further pursue. First, in the current work we only use a simple target prior model in the form of GB-RBM. There are, however, more flexible probabilistic models, such as Markov random fields or dynamic Bayesian networks, that can better represent statistical properties in the target variables. Therefore, we would like to incorporate such models into LTP learning to further improve the performance. Second, we would like to investigate the connection between conventional capacity control methods (e.g., max margin or regularization) with LTP learning. This has the potential to unify and shed light on the deeper relation among different learning methodologies. Last, we would also like to use LTP learning with nonlinear decoding functions.

**Acknowledgement** The authors would like to thank Jixu Chen for providing the motion capture data and feature extraction code. Zuoguan Wang and Qiang Ji are supported in part by a grant from US Army Research Office (W911NF-08-1-0216 (GS)) through Albany Medical College. Gerwin Schalk is supported by US Army Research Office (W911NF-08-1-0216 (GS)) and W911NF-07-1-0415 (GS), and the NIH (EB006356(GS) and EB000856 (GS)). Siwei Lyu is supported by an NSF CAREER Award (IIS-0953373).

# References

[1] Bashashati, Ali, Fatourechi, Mehrdad, Ward, Rabab K., and Birch, Gary E. A survey of signal processing algorithms in brain-computer interfaces based on electrical brain signals. *J. Neural Eng.*, 4, June 2007.

[2] Bougrain, Laurent and Liang, Nanying. Band-specific features improve Finger Flexion Prediction from ECoG. In *Jornadas Argentinas sobre Interfaces Cerebro Computadora - JAICC*, Paranà, Argentine, 2009.

[3] Chang, Mingwei, Ratinov, Lev, and Roth, Dan. Guiding semi-supervision with constraint-driven learning. In *Proc. of the Annual Meeting of the ACL*, 2007.

[4] Flamary, Rémi and Rakotomamonjy, Alain. Decoding finger movements from ECoG signals using switching linear models. Technical report, September 2009.

[5] Ganchev, Kuzman, Graca, Joao, Gillenwater, Jennifer, and Taskar, Ben. Posterior regularization for structured latent variable models. *JMLR*, 11(July):2001–2049, 2010.

[6] Hinton, Geoffrey. Training products of experts by minimizing contrastive divergence. *Neural Computation*, 14(8):2002, Aug 2000.

[7] Krusienski, Dean J, Grosse-Wentrup, Moritz, Galn, Ferran, Coyle, Damien, Miller, Kai J, Forney, Elliott, and Anderson, Charles W. Critical issues in state-of-the-art brain-computer interface signal processing. *Journal of Neural Engineering*, 8(2):025002, 2011.

[8] Kubánek, J, Miller, K J, Ojemann, J G, Wolpaw, J R, and Schalk, G. Decoding flexion of individual fingers using electrocorticographic signals in humans. *J Neural Eng*, 6(6):066001–066001, Dec 2009.

[9] Lafferty, John. Conditional random fields: Probabilistic models for segmenting and labeling sequence data. In *NIPS*, pp. 282–289. Morgan Kaufmann, 2001.

[10] Lefort, Riwal, Fablet, Ronan, and Boucher, Jean-Marc. Weakly supervised classification of objects in images using soft random forests. In *ECCV*, pp. 185–198, 2010.

[11] Liang, Percy, Jordan, Michael I., and Klein, Dan. Learning from measurements in exponential families. In *ICML '09*, pp. 641–648, New York, NY, USA, 2009. ACM.

[12] Mann, Gideon S. and McCallum, Andrew. Simple, robust, scalable semi-supervised learning via expectation regularization. In *ICML*, pp. 593–600, 2007.

[13] Mann, Gideon S. and Mccallum, Andrew. Generalized expectation criteria for semi-supervised learning of conditional random fields. In *ACL'08*, pp. 870–878, 2008.

[14] Mohamed, A., Dahl, G., and Hinton, G. Acoustic modeling using deep belief networks. *Audio, Speech, and Language Processing, IEEE Transactions on*, PP(99):1, 2011.

[15] Salzmann, Mathieu, Henrik, Carl, Raquel, Ek, and Darrell, Urtasun Trevor. Factorized orthogonal latent spaces. *JMLR*, 9:701–708, 2010.

[16] Schapire, Robert E., Rochery, Marie, Rahim, Mazin G., and Gupta, Narendra. Incorporating prior knowledge into boosting. In *ICML*, 2002.

[17] Shenoy, P., Miller, K.J., Ojemann, J.G., and Rao, R.P.N. Generalized features for electrocorticographic bcis. *Biomedical Engineering, IEEE Transactions on*, 55(1), jan. 2008.

[18] Shenoy, Pradeep, Krauledat, Matthias, Blankertz, Benjamin, Rao, Rajesh P. N., and Müller, Klaus-Robert. Towards adaptive classification for BCI. *Journal of Neural Engineering*, 2006.

[19] Shon, Aaron P., Grochow, Keith, Hertzmann, Aaron, and Rao, Rajesh P. N. Learning shared latent structure for image synthesis and robotic imitation. In *NIPS*, pp. 1233–1240, 2006.

[20] Tarlow, Daniel and S. Zemel, Richard. Structured output learning with high order loss functions. *AISTATS*, 2012.

[21] Taskar, Ben, Guestrin, Carlos, and Koller, Daphne. Max-margin markov networks. In *NIPS*. MIT Press, 2003.

[22] Taylor, G.W., Sigal, L., Fleet, D.J., and Hinton, G.E. Dynamical binary latent variable models for 3d human pose tracking. In *CVPR*, pp. 631 –638, June 2010.

[23] Tian, Tai-Peng, Li, Rui, and Sclaroff, S. Articulated pose estimation in a learned smooth space of feasible solutions. In *CVPR*, pp. 50, June 2005.

[24] Wang, Yijun and Jung, Tzyy-Ping. A collaborative brain-computer interface for improving human performance. *PLoS ONE*, 6(5):e20422, 05 2011.

[25] Wang, Zuoguan, Ji, Qiang, Miller, Kai J., and Schalk, Gerwin. Decoding finger flexion from electrocorticographic signals using a sparse gaussian process. In *ICPR*, pp. 3756–3759, 2010.

[26] Wang, Zuoguan, Schalk, Gerwin, and Ji, Qiang. Anatomically constrained decoding of finger flexion from electrocorticographic signals. In *NIPS*, 2011.

[27] Yu, C.-N. and Joachims, T. Learning structural SVMs with latent variables. In *ICML*, 2009.

[28] Zhu, Xiaojin. Semi-supervised learning literature survey, 2006. URL http://pages.cs.wisc.edu/~jerryzhu/pub/ssl_survey.pdf.

